# Bayesian inference in spiking neurons

**Sophie Deneve**[*]
Gatsby Computational Neuroscience Unit
University College London
London, UK WC1N 3AR
sdeneve@gatsby.ucl.ac.uk

## Abstract

We propose a new interpretation of spiking neurons as Bayesian integrators accumulating evidence over time about events in the external world or the body, and communicating to other neurons their certainties about these events. In this model, spikes signal the occurrence of new information, i.e. what cannot be predicted from the past activity. As a result, firing statistics are close to Poisson, albeit providing a deterministic representation of probabilities. We proceed to develop a theory of Bayesian inference in spiking neural networks, recurrent interactions implementing a variant of belief propagation.

Many perceptual and motor tasks performed by the central nervous system are probabilistic, and can be described in a Bayesian framework [4, 3]. A few important but hidden properties, such as direction of motion, or appropriate motor commands, are inferred from many noisy, local and ambiguous sensory cues. These evidences are combined with priors about the sensory world and body. Importantly, because most of these inferences should lead to quick and irreversible decisions in a perpetually changing world, noisy cues have to be integrated on-line, but in a way that takes into account unpredictable events, such as a sudden change in motion direction or the appearance of a new stimulus.

This raises the question of how this temporal integration can be performed at the neural level. It has been proposed that single neurons in sensory cortices represent and compute the log probability that a sensory variable takes on a certain value (*eg* Is visual motion in the neuron's preferred direction?) [9, 7]. Alternatively, to avoid normalization issues and provide an appropriate signal for decision making, neurons could represent the log probability ratio of a particular hypothesis (*eg* is motion more likely to be towards the right than towards the left) [7, 6]. Log probabilities are convenient here, since under some assumptions, independent noisy cues simply combine linearly. Moreover, there are physiological evidence for the neural representation of log probabilities and log probability ratios [9, 6, 7].

However, these models assume that neurons represent probabilities in their firing rates. We argue that it is important to study how probabilistic information are encoded in spikes. Indeed, it seems spurious to marry the idea of an exquisite on-line integration of noisy cues with an underlying rate code that requires averaging on large populations of noisy neurons and long periods of time. In particular, most natural tasks require this integration to take place on the time scale of inter-spike intervals. Spikes are more efficiently signaling events

---

[*]Institute of Cognitive Science, 69645 Bron, France

than analog quantities. In addition, a neural theory of inference with spikes will bring us closer to the physiological level and generate more easily testable predictions.

Thus, we propose a new theory of neural processing in which spike trains provide a *deterministic*, *online* representation of a log-probability ratio. Spikes signals *events*, *eg* that the log-probability ratio has exceeded what could be predicted from previous spikes. This form of coding was loosely inspired by the idea of "energy landscape" coding proposed by Hinton and Brown [2]. However, contrary to [2] and other theories using rate-based representation of probabilities, this model is self-consistent and does not require different models for encoding and decoding: As output spikes provide new, unpredictable, temporally independent evidence, they can be used directly as an input to other Bayesian neurons.

Finally, we show that these neurons can be used as building blocks in a theory of approximate Bayesian inference in recurrent spiking networks. Connections between neurons implement an underlying Bayesian network, consisting of coupled hidden Markov models. Propagation of spikes is a form of belief propagation in this underlying graphical model.

Our theory provides computational explanations of some general physiological properties of cortical neurons, such as spike frequency adaptation, Poisson statistics of spike trains, the existence of strong local inhibition in cortical columns, and the maintenance of a tight balance between excitation and inhibition. Finally, we discuss the implications of this model for the debate about temporal versus rate-based neural coding.

# 1   Spikes and log posterior odds

## 1.1   Synaptic integration seen as inference in a hidden Markov chain

We propose that each neuron codes for an underlying "hidden" binary variable, $x_t$, whose state evolves over time. We assume that $x_t$ depends only on the state at the previous time step, $x_{t-dt}$, and is conditionally independent of other past states. The state $x_t$ can switch from 0 to 1 with a constant rate $r_{\text{on}} = \frac{1}{dt} \lim_{dt \to 0} P(x_t = 1 | x_{t-dt} = 0)$, and from 1 to 0 with a constant rate $r_{\text{off}}$. For example, these transition rates could represent how often motion in a preferred direction appears the receptive field and how long it is likely to stay there.

The neuron infers the state of its hidden variable from $N$ noisy synaptic inputs, considered to be *observations* of the hidden state. In this initial version of the model, we assume that these inputs are conditionally independent homogeneous Poisson processes, synapse $i$ emitting a spike between time $t$ and $t + dt$ ($s_t^i = 1$) with constant probability $q_{\text{on}}^i dt$ if $x_t = 1$, and another constant probability $q_{\text{off}}^i dt$ if $x_t = 0$. The synaptic spikes are assumed to be otherwise independent of previous synaptic spikes, previous states and spikes at other synapses. The resulting generative model is a hidden Markov chain (figure 1-A).

However, rather than *estimating* the state of its hidden variable and communicating this estimate to other neurons (for example by emitting a spike when sensory evidence for $x_t = 1$ goes above a threshold) the neuron reports and communicates its *certainty* that the current state is 1. This certainty takes the form of the log of the ratio of the probability that the hidden state is 1, and the probability that the state is 0, given all the synaptic inputs received so far: $L_t = \log\left(\frac{P(x_t=1|\mathbf{s}_{0 \to t})}{P(x_t=0|\mathbf{s}_{0 \to t})}\right)$. We use $\mathbf{s}_{0 \to t}$ as a short hand notation for the $N$ synaptic inputs received at present *and in the past*. We will refer to it as the log odds ratio.

Thanks to the conditional independencies assumed in the generative model, we can compute this Log odds ratio iteratively. Taking the limit as $dt$ goes to zero, we get the following differential equation:

$$\dot{L} = r_{\text{on}} \left(1 + e^{-L}\right) - r_{\text{off}} \left(1 + e^{L}\right) + \sum_i w_i \delta(s_t^i - 1) - \theta$$

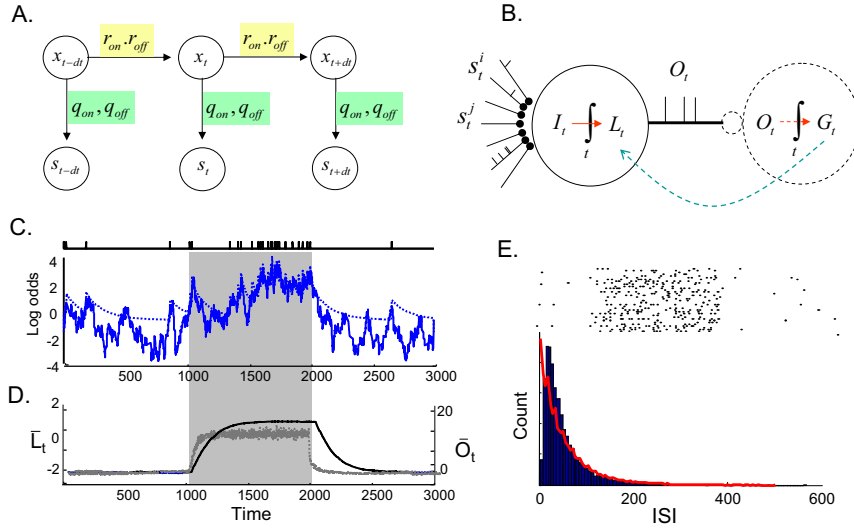

Figure 1: A. Generative model for the synaptic input. B. Schematic representation of log odds ratio encoding and decoding. The dashed circle represents both eventual downstream elements and the self-prediction taking place *inside* the model neuron. A spike is fired only when $L_t$ exceeds $G_t$. C. One example trial, where the state switches from 0 to 1 (shaded area) and back to 0. plain: $L_t$, dotted: $G_t$. Black stripes at the top: corresponding spikes train. D. Mean Log odds ratio (dark line) and mean output firing rate (clear line). E. Output spike raster plot (1 line per trial) and ISI distribution for the neuron shown is C. and D. Clear line: ISI distribution for a poisson neuron with the same rate.

$w_i$, the synaptic weight, describe how informative synapse $i$ is about the state of the hidden variable, *e.g.* $w_i = \log\left(\frac{q^i_{\text{on}}}{q^i_{\text{off}}}\right)$. Each synaptic spike ($s^i_t = 1$) gives an impulse to the log odds ratio, which is positive if this synapse is more active when the hidden state if 1 (i.e it increases the neuron's confidence that the state is 1), and negative if this synapse is more active when $x_t = 0$ (i.e it decreases the neuron's confidence that the state is 1).

The bias, $\theta$, is determined by how informative it is *not* to receive any spike, *e.g.* $\theta = \sum_i q^i_{\text{on}} - q^i_{\text{off}}$. By convention, we will consider that the "bias" is positive or zero (if not, we need simply to invert the status of the state $x$).

## 1.2  Generation of output spikes

The spike train should convey a sparse representation of $L_t$, so that each spike reports *new information* about the state $x_t$ that is not redundant with that reported by other, preceding, spikes. This proposition is based on three arguments: First, spikes, being metabolically expensive, should be kept to a minimum. Second, spikes conveying redundant information would require a decoding of the entire spike train, whereas independent spike can be taken into account individually. And finally, we seek a self consistent model, with the spiking output having a similar semantics to its spiking input.

To maximize the independence of the spikes (conditioned on $x_t$), we propose that the neuron fires only when the difference between its log odds ratio $L_t$ and a *prediction* $G_t$ of this log odds ratio based on the output spikes emitted so far reaches a certain threshold. Indeed, supposing that downstream elements predicts $L_t$ as best as they can, the neuron only needs to fire when it expects that prediction to be too inaccurate (figure 1-B). In practice, this

will happen when the neuron receives new evidence for $x_t = 1$. $G_t$ should thereby follow the same dynamics as $L_t$ when spikes are not received. The equation for $G_t$ and the output $O_t$ ($O_t = 1$ when an output spike is fired) are given by:

$$\dot{G} = r_{\text{on}}\left(1 + e^{-L}\right) - r_{\text{off}}\left(1 + e^{L}\right) + g_o\delta(O_t - 1) \tag{1}$$

$$O_t = 1. \text{ when } L_t > G_t + \frac{g_o}{2}, 0 \text{ otherwise}, \tag{2}$$

Here $g_o$, a positive constant, is the only free parameter, the other parameters being constrained by the statistics of the synaptic input.

## 1.3 Results

Figure 1-C plots a typical trial, showing the behavior of $L$, $G$ and $O$ before, during and after presentation of the stimulus. As random synaptic inputs are integrated, $L$ fluctuates and eventually exceeds $G + 0.5$, leading to an output spike. Immediately after a spike, $G$ jumps to $G + g_o$, which prevents (except in very rare cases) a second spike from immediately following the first. Thus, this "jump" implements a relative refractory period. However, $G$ decays as it tends to converge back to its stable level $g_{\text{stable}} = \log\left(\frac{r_{\text{on}}}{r_{\text{off}}}\right)$. Thus $L$ eventually exceeds $G$ again, leading to a new spike. This threshold crossing happens more often during stimulation ($x_t = 1$) as the net synaptic input alters to create a higher overall level of certainty, $L_t$.

*Mean Log odds ratio and output firing rate*

The mean firing rate $\bar{O}_t$ of the Bayesian neuron during presentation of its preferred stimulus (i.e. when $x_t$ switches from 0 to 1 and back to 0) is plotted in figure 1-D, together with the mean log posterior ratio $\bar{L}_t$, both averaged over trials. Not surprisingly, the log-posterior ratio reflects the leaky integration of synaptic evidence, with an effective time constant that depends on the transition probabilities $r_{\text{on}}, r_{\text{off}}$. If the state is very stable ($r_{\text{on}} = r_{\text{off}} \sim 0$), synaptic evidence is integrated over almost infinite time periods, the mean log posterior ratio tending to either increase or decrease linearly with time. In the example in figure 1-D, the state is less stable, so "old" synaptic evidence are discounted and $L_t$ saturates.

In contrast, the mean output firing rate $\bar{O}_t$ tracks the state of $x_t$ almost perfectly. This is because, as a form of predictive coding, the output spikes reflect the *new* synaptic evidence, $I_t = \sum_i \delta(s_t^i - 1) - \theta$, rather than the log posterior ratio itself. In particular, the mean output firing rate is a rectified linear function of the mean input, *e. g.* $\bar{O} = \frac{1}{g_o}\bar{I} = \left[\sum_i w_i q_{\text{on(off)}}^i - \theta\right]^+$.

*Analogy with a leaky integrate and fire neuron*

We can get an interesting insight into the computation performed by this neuron by linearizing $L$ and $G$ around their mean levels over trials. Here we reduce the analysis to prolonged, statistically stable periods when the state is constant (either ON or OFF). In this case, the mean level of certainty $\bar{L}$ and its output prediction $\bar{G}$ are also constant over time. We make the rough approximation that the post spike jump, $g_o$, and the input fluctuations are small compared to the mean level of certainty $\bar{L}$.

Rewriting $V_t = L_t - G_t + \frac{g_o}{2}$ as the "membrane potential" of the Bayesian neuron:

$$\dot{V} = -k_{\bar{L}}V + I_t - \Delta_{g_o} - g_oO_t$$

where $k_{\bar{L}} = r_{\text{on}}e^{-\bar{L}} + r_{\text{off}}e^{\bar{L}}$, the "leak" of the membrane potential, depends on the overall level of certainty. $\Delta_{g_o}$ is positive and a monotonic increasing function of $g_o$.

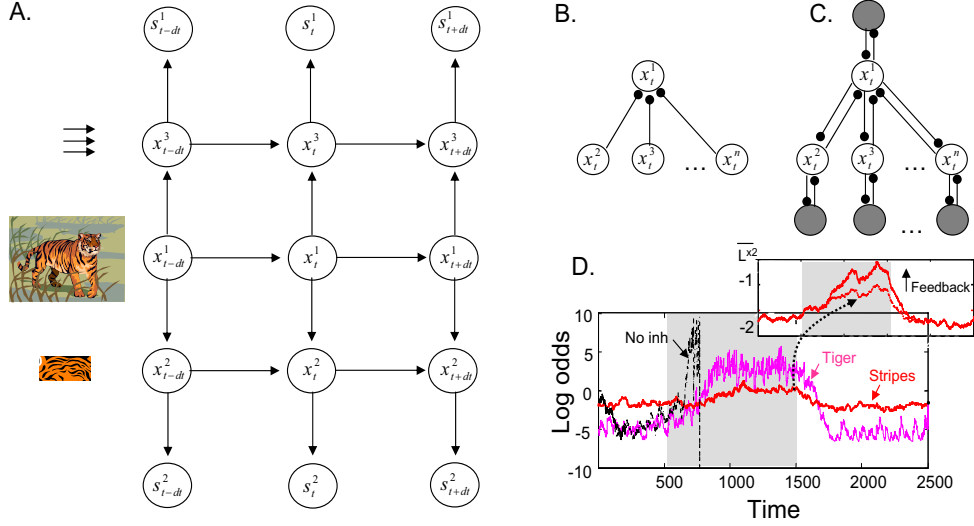

Figure 2: A. Bayesian causal network for $y_t$ (tiger), $x_t^1$ (stripes) and $x_t^2$ (paws). B. A network feedforward computing the log posterior for $x_t^1$. C. A recurrent network computing the log posterior odds for all variables. D. Log odds ratio in a simulated trial with the network in C (see text). Thick line: $L_t^{x^2}$, thin line: $L_t^{x^1}$, dash-dotted: $L_t^{x^1}$ without inhibition. Insert: $L_t^{x^2}$ averaged over trials, showing the effect of feedback.

The linearized Bayesian neuron thus acts in its stable regime as a leaky integrate and fire (LIF) neuron. The membrane potential $V_t$ integrates its input, $J_t = I_t - \Delta_{g_o}$, with a leak $k_{\bar{L}}$. The neuron fires when its membrane potential reaches a constant threshold $g_o$. After each spikes, $V_t$ is reset to $0$.

Interestingly, for appropriately chosen compression factor $g_o$, the mean input to the linearized neuron $\bar{J} = \bar{I} - \Delta_{g_o} \approx 0$ [1]. This means that the membrane potential is purely driven to its threshold by input fluctuations, or a random walk in membrane potential. As a consequence, the neuron's firing will be memoryless, and close to a Poisson process. In particular, we found Fano factor close to 1 and quasi-exponential ISI distribution (figure 1-E) on the entire range of parameters tested. Indeed, LIF neurons with balanced inputs have been proposed as a model to reproduce the statistics of real cortical neurons [8]. This balance is implemented in our model by the neuron's effective self-inhibition, even when the synaptic input itself is not balanced.

*Decoding*

As we previously said, downstream elements could predict the log odds ratio $L_t$ by computing $G_t$ from the output spikes (Eq 1, fig 1-B). Of course, this requires an estimate of the transition probabilities $r_{\mathrm{on}}, r_{\mathrm{off}}$, that could be learned from the observed spike trains.

However, we show next that explicit decoding is not necessary to perform bayesian inference in spiking networks. Intuitively, this is because the quantity that our model neurons receive and transmit, *eg* new information, is exactly what probabilistic inference algorithm propagate between connected statistical elements.

## 2 Bayesian inference in cortical networks

The model neurons, having the same input and output semantics, can be used as building blocks to implement more complex generative models consisting of coupled Markov chains. Consider, for example, the example in figure 2-A. Here, a "parent" variable $x_t^1$ (the presence of a tiger) can cause the state of $n$ other "children" variables ($[x_t^k]_{k=2...n}$), of whom two are represented (the presence of stripes,$x_t^2$, and motion, $x_t^3$). The "children" variables are Bayesian neurons identical to those described previously. The resulting bayesian network consist of $n+1$ coupled hidden Markov chains. Inference in this architecture corresponds to computing the log posterior odds ratio for the tiger, $x_t^1$, *and* the log posterior of observing stripes or motion, ($[x_t^k]_{k=2...n}$), given the synaptic inputs received by the *entire* network so far, i.e. $\mathbf{s}_{0\rightarrow t}^2, \ldots, \mathbf{s}_{0\rightarrow t}^k$.

Unfortunately, inference and learning in this network (and in general in coupled Markov chains) requires very expensive computations, and cannot be performed by simply propagating messages over time and among the variable nodes. In particular, the state of a child variable $x_t^k$ depends on $x_{t-dt}^k$, $\mathbf{s}_t^k$, $x_t^1$ *and* the state of all other children at the previous time step, $[x_{t-dt}^j]_{2<j<n, j\neq i}$. In contrast, our network can only implement pairwise interactions, a connection between two spiking neurons implementing the conditional probability linking the two corresponding binary variables.

Thus, we need to assume additional conditional independencies between the nodes in the generative model, so that their joint probability can be pairwise factorized: $p(\mathbf{x}_t, \mathbf{x}_{t-1}) = \frac{1}{Z}\prod_{ij}\phi(x_t^i, x_t^j)\prod_i \phi(x_t^i, x_{t-dt}^i)$. In words, it means that variables bias each other's probabilities, but do not influence each other's dynamics, i.e they do not affect each other's transition probabilities. For example, a tiger does not affect the probability that stripes appear or disappear, but increases their probability of being present.

*Naive implementation*

In this restricted case, marginal posterior probabilities can be computed iteratively by propagating beliefs in time and between the variables, or, in our model, by propagating spikes in a neural network. This is because the probability of a variable $x_t^k$ can be directly updated by the conditional probability of observing the synaptic input to another connected neuron, $s_t^l$, *eg* $p(s_t^l|x_t^k) = \sum_{x_t^l} p(s_t^l|x_t^l)p(x_t^l|x_t^k)$, marginalizing out the hidden state $x^l$. Of course, rather than using $s_t^l$, we use $O_t^l$, the output of the Bayesian neuron coding for $x_t^l$. As we said previously, this output directly represents the new synaptic evidence received by neuron $l$. The resulting equation is identical to the one derived previously for poisson input,

$$\dot{L}^{x^k} = f_k(L^{x^k}) + \sum_l w_{lk}\delta(O_t^l - 1) - \theta_k$$

where $f_k(x) = r_{\text{on}}^k(1 + e^{-x}) - r_{\text{off}}^k(1 + e^x)$. As previously, $w_{lk}$, the synaptic weight, describes how informative it is for neuron $k$ to receive a spike from neuron (or synapse) $l$, $w_{lk} = \log(\frac{P(O_t^k=1|y_t=1)}{P(O_t^k=1|y_t=0)})$, while $\theta_k$ is how informative it is not to receive a spike, $\theta_k = dt\sum_l P(O_t^l = 1|x_t^k = 1) - P(O_t^l = 1|x_t^k = 0)$.

This shows that our model is self-consistent. Except at the first stage of processing (*eg* in the retina), all inputs are proposed to come from other Bayesian neurons.

*Results*

We implemented these update rules in a spiking neural network (figure 2-B) representing the generative model in figure 2-A, with 100 possible children for $x_t^1$. We first consider the case where there is no feedback connections, meaning that $w_{1k} = 0$ for all $k$. In this case the network computes the probability of a tiger at time $t$, integrating multiple sensory cues such as the presence of stripes or motion in the visual scene.

In the example trials plotted in figure 2-D, we fixed the state of $x_t^1$ and $x_t^2$: the tiger and the stripes are present in the shaded temporal window, and absent outside of it. We then sample the states of the other children (i.e is there motion or not?) and the corresponding "observed" synaptic inputs, $\mathbf{s}_{0 \to t_{\max}}^k$, from the generative model. Once this synaptic input has been generated, it is used as an input to the network in figure 2-B. What is plotted in the Log odds ratio for the tiger, $L_t^{x_1}$ and the stripes, $L_t^{x_1}$, as a function of time. As we can see, the stripes receive very noisy synaptic input and can only provide weak evidence that they are present. However, the tiger neuron is able to combine inputs from its 100 children and get a much higher certainty.

Unfortunately, bayesian inference in this feedforward network is incomplete: The presence of a tiger affects the probability of stripes, not only the other way round. To implement this, we also need feedback connections, $w_{1k}$. The network with feedforward and feedback processing fails miserably, given that its activity explode, as illustrated in figure 2-D.

*Balanced excitation/inhibition*

This failure is due to the presence of loops, whereby a spike from neuron $k$ increases the certainty of neuron $l$ (and its probability of firing) by $w_{kl}$, and a spike from neuron $l$ increases in turn the certainty of neuron $k$ by $w_{lk}$. These loops result in spikes reverberating through the network, ad infinitum, without reporting new information, a phenomena akin to loopy belief propagation [10]. To avoid overcounting of evidence, we thus have to discount the reverberated "old evidence" from the synaptic input:

$$\dot{L}^{x^k} = f_k(L^{x^k}) + \sum_l w_{kl}\delta(O_t^l - 1) - \sum_l w_{kl}w_{lk}\delta(O_{t-dt}^k - 1) - \theta_k.$$

We implemented this discounting using inhibitory neurons recurrently connected to each excitatory neuron (figure 2-C). The inhibitory neurons are used to *predict* the redundant feedback a bayesian neuron will receive and substract this prediction so that, once again, *only new information* are taken into account and communicated to other neurons. Each excitatory loop is compensated by an inhibitory loop, resulting in a balance between excitation and inhibition at the level of each neuron within the network.

The result on one trial is plotted in figure 2-D. The "tiger" log odds ratio is almost indistinguishable from the feedforward case, and is not plotted. The "stripes" Log odds ratio increases during presentation of the tiger due to the feedback. In other words, the stripes neuron can take into account no only its own its own synaptic input, but also the synaptic input to the other children neurons (such as evidence for motion), thanks to the presence of a common source (the tiger).

Over many trials, we found that the statistics of the bayesian neuron were still poisson, and their output firing is still a rectified linear function of the input firing rate in a stable statistical regime, i.e. $\bar{O}^k = [\sum_l w_{kl}\bar{O}^l]^+$.

## Discussion

We started from an interpretation of synaptic integration in single neurons as a form of inference in a hidden Markov chain. We derived a model of spiking neurons and their interactions able to compute the marginal posterior probabilities of sensory and motor variables given evidence received *in the entire network*. In this view, the brain implements an underlying bayesian network in an interconnected neural architecture, with conditional probabilities represented by synaptic weights. The model makes a rich set of predictions for the general properties of neuron and synaptic dynamics, such as a time constant that depends on the overall level of inputs, specific forms of frequency dependant spike and synaptic adaptation (not shown here) and micro-balanced excitation and inhibition. However, it is still restricted to probabilistic computations involving binary variables. In a

related work similar ideas are applied population encoding of log probability distribution for analog variables (Zemel, Huys and Dayan, submitted to NIPS 2004).

Despite non-linear processing at the single neural level, the emerging picture is relatively simple: The neuron acts as a leaky integrate and fire neuron driven by noise. The output firing rate is a rectified weighted sum of the input firing rates, while the firing statistics are Poisson. However, these output spike trains are a *deterministic* function of the input spike trains. Spikes report fluctuations in the level of certainty *that could not be predicted* either from the stability of its stimulus (contribution from $G_t$) or the loops in the network (contribution from the inhibitory neuron). Thus firing will be, by definition, unpredictable. This last observation leads us to suggest that the irregular firing and Poisson statistics observed in cortical neurons [1] arises as a direct consequence of the random fluctuations in the sensory inputs and the instability of the real word, but are *not* due to unreliable or "chaotic" neural processing.

Finally, it is crucial for the biological realism of the model to find adaptive neural dynamics and synaptic plasticity able to learn the parameters of the internal model and conditional probabilities, and we are currently exploring these issues. Fortunately, the required learning rules are local and unsupervised. According to our preliminary work, the synaptic weights and bias depend on the joint probability of presynaptic/postsynaptic spikes and can be learned with the spike time dependent plasticity observed in hippocampus and cortex [5]. Meanwhile, the transition probabilities simply correspond to how often the neuron switches between an active and an inactive state.

### Acknowledgments

We thank Peter Dayan, Peter Latham, Zoubin Ghahramani, Jean Laurens and Jacques Droulez for helpful discussions and comments. This work was supported by a Dorothy Hodgkin fellowship for the Royal Society and the BIBA European Project.

## Footnotes

[1]Even if $g_o$ is not chosen optimally, the influence of the drift $\bar{J}$ is usually negligible compared to the large fluctuations in membrane potential.

# References

[1] K. H. Britten, M. N. Shadlen, W. T. Newsome, and J. A. Movshon. The analysis of visual motion: A comparison of neuronal and psychophysical performance. *Journal of Neuroscience*, 12:4745–4765, 1992.

[2] G. Hinton and A. Brown. Spiking boltzmann machines. In S. Solla, T. Leen, and K. Muller, editors, *Neural Information Processing System*, volume 12, pages 122–8. MIT Press, Cambridge, MA, 2000.

[3] D. Knill and W. Richards. *Perception as Bayesian inference*. Cambridge University Press, Cambridge, MA, 1996.

[4] K. Kording and D. Wolpert. Bayesian integration in sensorimotor learning. *Nature*, 427:244–7, 2004.

[5] H. Markram and M. Tsodyks. Redistribution of synaptic efficacy between neocortical pyramidal neurons. *Nature*, 382:807–19, 1996.

[6] M. Mazurek, J. Roitman, J. Ditterich, and M. Shadlen. A role for neural integrators in perceptual decision making. *Cerebral cortex*, 13(11):1257–69, 2003.

[7] R. Rao. Bayesian computation in recurrent neural circuits. *Neural Computation*, 16(1):1–38, 2003.

[8] M. Shadlen and W. Newsome. Noise, neural codes and cortical organization. *Current Opinion in Neurobiology*, 4:569–579, 1994.

[9] Y. Weiss and D. Fleet. Velocity likelihood in biological and machine vision. In R. Rao, B. Olshausen, and M. Lewicki, editors, *Probabilistic Models of the Brain: Perception and Neural Function*, pages 77–96. MIT Press, Cambridge, MA, 2002.

[10] Y. Weiss and W. Freeman. Correctness of belief propagation in gaussian graphical models of arbitrary topology. *Neural Computation*, 13:2173–200, 2001.
